# TAP Gibbs Free Energy, Belief Propagation and Sparsity

**Lehel Csató** and **Manfred Opper**
Neural Computing Research Group
School of Engineering and Applied Science
Aston University, Birmingham B4 7ET, UK.
`[csatol,opperm]@aston.ac.uk`

**Ole Winther**
Center for Biological Sequence Analysis, BioCentrum
Technical University of Denmark, B208, 2800 Lyngby, Denmark.
`winther@cbs.dtu.dk`

## Abstract

The adaptive TAP Gibbs free energy for a general densely connected probabilistic model with quadratic interactions and arbitrary single site constraints is derived. We show how a specific sequential minimization of the free energy leads to a generalization of Minka's expectation propagation. Lastly, we derive a sparse representation version of the sequential algorithm. The usefulness of the approach is demonstrated on classification and density estimation with Gaussian processes and on an independent component analysis problem.

## 1 Introduction

There is an increasing interest in methods for approximate inference in probabilistic (graphical) models. Such approximations may usually be grouped in three classes. In the first case we approximate *self-consistency relations* for marginal probabilities by a set of nonlinear equations. Mean field (MF) approximations and their advanced extensions belong to this group. However, it is not clear in general, how to solve these equations efficiently. This latter problem is of central concern to the second class, the *Message passing algorithms*, like Bayesian online approaches (for references, see e.g. [1]) and belief propagation (BP) which dynamically update approximations to conditional probabilities. Finally, approximations based on *Free Energies* allow us to derive marginal moments by minimising entropic loss measures. This method introduces new possibilities for algorithms and also gives approximations for the log-likelihood of observed data. The *variational method* is the most prominent member of this group.

One can gain important insight into an approximation, when it can be derived by different approaches. Recently, the fixed points of the BP algorithm were identified as the stable minima of the *Bethe* Free Energy, an insight which led to improved approximation schemes [2]. While BP is good and efficient on sparse tree-like structures, one may look for an approxi-

mation that works well in the opposite limit of densely connected graphs where individual dependencies are weak but their overall effect cannot be neglected. A interesting candidate is the adaptive TAP (ADATAP) approach introduced in [3] as a set of self-consistency relations. Recently, a message passing algorithm of Minka (termed *expectation propagation*) [1] was found to solve the ADATAP equations efficiently for models with Gaussian Process (GP) priors.

The goal of this paper is three-fold. We will add a further derivation of ADATAP using an approximate free energy. A sequential algorithm for minimising the free energy generalises Minka's result. Finally, we discuss how a sparse representation of ADATAP can be achieved for GP models, thereby extending previous sparse on-line approximations to the batch case [4].

We will specialize to probabilistic models on undirected graphs with nodes $i$ that are of the type

$$P_\rho(\mathbf{S}) = \frac{\rho(\mathbf{S})}{Z} \exp\left[\sum_{i<j} S_i J_{ij} S_j\right] \tag{1}$$

The set of $J_{ij}$'s encodes the dependencies between the random variables $\mathbf{S} = (S_1, \ldots, S_N)$, whereas the factorising term $\rho(\mathbf{S}) = \prod_j \rho_j(S_j)$ (called *likelihood* in the following) usually encodes observed data at sites $i$ and also incorporates all local constraints of the $S_i$ (the range, discreteness, etc). Hence, depending on these constraints, $S_i$ maybe discrete or continuous. Eq. (1) is a sufficiently rich and interesting class of models containing Boltzmann machines, models with Gaussian process priors [3], probabilistic independent component analysis [5] as well as Bayes belief networks and probabilistic neural networks (when the space of variables is augmented by auxiliary integration variables).

## 2 ADATAP approach from Gibbs Free Energy

We use the minimization of an approximation to a *Gibbs Free Energy* $G$ in order to re-derive the ADATAP approximation.

The Gibbs Free Energy provides a method for computing marginal moments of $P$ as well as of $-\ln Z$ within the same approach. It is defined by a constrained relative entropy minimization which is, for the present problem defined as

$$G_\rho(\mathbf{m}, \mathbf{M}) = \min_Q \left\{ KL(Q, P_\rho) \mid \langle \mathbf{S} \rangle_Q = \mathbf{m}, \langle \mathbf{S}^2 \rangle_Q = \mathbf{M} \right\} - \ln Z , \tag{2}$$

where the brackets denote expectations with respect to the distribution $Q$ and $\langle \mathbf{S}^2 \rangle_Q$ is shorthand for a vector with elements $\langle S_i^2 \rangle_Q$. Finally, $KL(Q, P_\rho) \doteq \int d\mathbf{S} \, Q(\mathbf{S}) \ln \frac{Q(\mathbf{S})}{P_\rho(\mathbf{S})}$. Since at the total minimum of $G$ (with respect to its arguments) the minimizer in (2) is just $Q = P_\rho$, we conclude that $\min_{\mathbf{m},\mathbf{M}} G(\mathbf{m}, \mathbf{M}) = -\ln Z$ and the desired marginal moments of $P$ are $(\langle \mathbf{S} \rangle, \langle \mathbf{S}^2 \rangle) = \operatorname{argmin}_{\mathbf{m},\mathbf{M}} G(\mathbf{m})$.

We will search for an approximation to $G_\rho$ which is based on splitting $G_\rho = G_\rho^0 + \Delta G_\rho$, where $G^0$ is the Gibbs free energy for a factorising model that is obtained from (1) by setting all $J_{ij} = 0$. Previous attempts [6, 7] were based on a truncation of the power series expansion of $\Delta G_\rho$ with respect to the $J_{ij}$ at second order. While this truncation leads to the correct TAP equations for the large $N$ limit of the so-called SK-model in statistical physics, its general significance is unclear. In fact, it will *not be exact* for a *simple* model with *Gaussian* likelihood. To make our approximation exact for such a case, we define (generalizing an idea of [8]) for an *arbitrary Gaussian* likelihood $\rho_i^g$, $\Delta G^g(\mathbf{m}, \mathbf{M}) \doteq G_{\rho^g}(\mathbf{m}, \mathbf{M}) - G_{\rho^g}^0(\mathbf{m}, \mathbf{M})$. The main reason for this definition is the fact that $\Delta G^g(\mathbf{m}, \mathbf{M})$ is *independent of* the actual Gaussian likelihood $\rho_i^g$ chosen to compute $G_\rho$! This result

depends crucially on the moment constraints in (2). Changes in a Gaussian likelihood can always be absorbed within the Lagrange-multipliers for the constraints. We use this *universal* form $\Delta G^g$ to define the ADATAP approximation as $G_\rho^{TAP} = G_\rho^0 + \Delta G^g$, which by construction is exact for any Gaussian likelihood $\rho$. Introducing appropriate Lagrange multipliers $\boldsymbol{\gamma}$ and $\boldsymbol{\lambda}$, we get

$$\Delta G^g(\mathbf{m}, \mathbf{M}) = \max_{\boldsymbol{\lambda}, \boldsymbol{\gamma}} \left\{ -\ln Z^g(\boldsymbol{\gamma}, \boldsymbol{\lambda}) + \mathbf{m}^T \boldsymbol{\gamma} - \frac{1}{2} \mathbf{M}^T \boldsymbol{\lambda} \right\} - \frac{1}{2} \sum_i \ln \left( M_i - m_i^2 \right) \quad (3)$$

with $Z^g(\boldsymbol{\gamma}, \boldsymbol{\lambda}) = \int d\mathbf{S} \ \exp\left[ \sum_i (\gamma_i S_i - \frac{1}{2}\lambda_i S_i^2) + \sum_{i<j} S_i J_{ij} S_j \right]$. Finally, setting $Z_i^0(\gamma_i^0, \lambda_i^0) = \int dS \rho_i(S) \ \exp[\gamma_i^0 S - \frac{1}{2}\lambda_i^0 S^2]$, we have

$$G_0 = \max_{\boldsymbol{\lambda}^0, \boldsymbol{\gamma}^0} \left\{ -\sum_i \ln Z_i^0(\gamma_i^0, \lambda_i^0) + \mathbf{m}^T \boldsymbol{\gamma}^0 - \frac{1}{2} \mathbf{M}^T \boldsymbol{\lambda}^0 \right\} . \quad (4)$$

## 3 Sequential Algorithm

The expression of $G_\rho^{TAP}$ in terms of moments $(\mathbf{m}, \mathbf{M})$ and Lagrange parameters $\boldsymbol{\gamma}, \boldsymbol{\lambda}$ and $\boldsymbol{\gamma}^0, \boldsymbol{\lambda}^0$ suggests that we may find local minima of $G_\rho^{TAP}$ by iteratively alternating between updates of moments and Lagrange multipliers. Of special interest is the following sequential algorithm, which is a generalization of Minka's EP [1] for Gaussian process classification to an arbitrary model of the type eq. (1).

We choose a site $i$ and define the updates by using the saddle points of $G_\rho$ with respect to the moments and Lagrange multipliers in the following sequential order (where $\boldsymbol{\Lambda}$ is a diagonal matrix with elements $\lambda_i$):

$$\begin{aligned}
\partial_{\gamma_i, \lambda_i} G_\rho = 0 &\quad \Rightarrow \quad m_i := \sum_j \left[ (\boldsymbol{\Lambda} - \mathbf{J})^{-1} \right]_{ij} \gamma_j &\& \quad M_i - m_i^2 := \left[ (\boldsymbol{\Lambda} - \mathbf{J})^{-1} \right]_{ii} \\
\partial_{m_i, M_i} G_\rho = 0 &\quad \Rightarrow \quad \gamma_i^0 := -\gamma_i - \frac{m_i}{M_i - m_i^2} &\& \quad \lambda_i^0 := -\lambda_i - \frac{1}{M_i - m_i^2} \\
\partial_{\gamma_i^0, \lambda_i^0} G_\rho = 0 &\quad \Rightarrow \quad m_i := \partial_{\gamma_i^0} \ln Z_i^0 &\& \quad M_i := -2 \partial_{\lambda_i^0} \ln Z_i^0 \\
\partial_{m_i, M_i} G_\rho = 0 &\quad \Rightarrow \quad \gamma_i := -\gamma_i^0 - \frac{m_i}{M_i - m_i^2} &\& \quad \lambda_i := -\lambda_i^0 - \frac{1}{M_i - m_i^2} .
\end{aligned}$$

The algorithm proceeds then by choosing a new site. The computation of $(\boldsymbol{\Lambda} - \mathbf{J})^{-1}$ can be performed efficiently using the Sherman-Woodbury formula because *only one element* $\lambda_i$ is changed in each update.

### 3.1 Cavity interpretation

At the fixed point, we may take $P_i(S) \equiv \frac{\rho_i(S)}{Z_i^0} \exp[\gamma_i^0 S - \frac{1}{2}\lambda_i^0 S^2]$ as the ADATAP approximation to the true marginal distribution of $S_i$ [3]. The sequential approach may thus be considered as a belief propagation algorithm for ADATAP.

Although $P_i$ is usually not Gaussian, we can also derive the moments $\mathbf{m}$ and $\mathbf{M}$ from the Gaussian distribution corresponding to $Z^g$. This auxiliary Gaussian model $P^g(\mathbf{S})$ has a likelihood $\rho_i^g(S) \propto \exp[-\frac{1}{2}\lambda_i S^2 + \gamma_i S]$ and provides us also with an additional approximation to the matrix of covariances via $\chi = (\boldsymbol{\Lambda} - \mathbf{J})^{-1}$. This is useful when the coupling matrix $\mathbf{J}$ must be adapted to a set of observations by maximum likelihood II. We will give an example of this for independent component analysis below.

It is important to understand the role of $\gamma^0$ and $\lambda^0$ within the "cavity" approach to the TAP equations. Defining $h_i = \sum_j J_{ij} S_j$, it is easy to show that $\gamma_i^0 = \langle h_i \rangle_{\backslash i}$ and $\lambda_i^0 = \langle h_i^2 \rangle_{\backslash i} - \langle h_i \rangle_{\backslash i}^2$ where the brackets denote an expectation with respect to the distribution of

all remaining variables $P^g(\mathbf{S} \setminus S_i) \propto \prod_{j, j \neq i} \rho^g_j(S_j) \exp[\sum_{k < l \neq i} J_{kl} S_k S_l]$ when node $i$ is deleted from the graph. This statistics of $h_i$ corresponds to the empty "cavity" at site $i$. The marginal distribution $P_i(S)$ as computed by ADATAP is equivalent to the approximation that the cavity distribution is Gaussian.

# 4 Examples

## 4.1 Models with Gaussian Process Priors

For this class of models, we assume that the graph is embedded in $R^D$, where the vector $\mathbf{S}$ is the restriction of a Gaussian process (random field) $\phi(\mathbf{x})$ with $\mathbf{x} \in R^D$, to a set of training inputs via $S_i = \phi(\mathbf{x}_i)$. $P_\rho(\mathbf{S})$ is the posterior distribution corresponding to a local likelihood model, when we set $\mathbf{J} = -\mathbf{K}^{-1}$ and the matrix $\mathbf{K}$ is obtained from a positive definite covariance kernel as $K_{ij} = K_0(\mathbf{x}_i, \mathbf{x}_j)$. The diagonal element $(K^{-1})_{ii}$ is included in the likelihood term.

Our ADATAP approximation can be extended from the finite set of inputs to the entire space $R^D$ by extending the auxiliary Gaussian distribution $P^g$ with its likelihoods $\rho^g_i(S)$ to a Gaussian process with mean $\langle \phi(\mathbf{x}) \rangle$ and posterior covariance kernel $K_p(\mathbf{x}, \mathbf{x}')$ which approximates the posterior process. A calculation similar to [4] leads to the representation

$$\langle \phi(\mathbf{x}) \rangle = \sum_j K_0(\mathbf{x}, \mathbf{x}_j) \gamma_j \tag{5}$$

$$K_p(\mathbf{x}, \mathbf{x}') = K_0(\mathbf{x}, \mathbf{x}') + \sum_{j,k} K_0(\mathbf{x}, \mathbf{x}_j) \chi_{jk} K_0(\mathbf{x}_k, \mathbf{x}') \tag{6}$$

Algorithms for the update of $\gamma$'s and $\chi$'s will usually suffer from time consuming matrix multiplications when $N$ is large. This common problem for GP models can be overcome by a *sparsity* approximation which extends previous on-line approaches [4] to the batch ADATAP approach. The idea is to replace the current version $P^g$ of the approximate Gaussian with a further approximation $\hat{P}^g$ for which both the the corresponding $\hat{\gamma}_j$ as well as $\hat{\chi}_{jk}$ are nonzero only, when the nodes $j$ and $k$ belong to a smaller subset of nodes called "basis vectors" (BV) of size $n$ [4]. For fixed BV set, the parameters of $\hat{P}^g$ are determined by minimizing the relative entropy $KL(\hat{P}^g, P^g)$. This yields $\hat{\gamma} = \boldsymbol{\pi} \gamma$ and $\hat{\mathbf{\Lambda}} = \boldsymbol{\pi} \mathbf{\Lambda} \boldsymbol{\pi}^T$ with the $n \times N$ projection matrix $\boldsymbol{\pi} = \mathbf{K}_{BV}^{-1} \mathbf{K}^+$. Here $\mathbf{K}$ is the kernel matrix between BVs and and $\mathbf{K}^+$ the kernel matrix between BVs and all nodes. The new distribution $\hat{P}^g$ can be written in the form (1) with a likelihood that contains only BVs

$$\hat{\rho}^g(\mathbf{S}^{BV}) = \exp[\sum_i \gamma_i (\boldsymbol{\pi}^T \mathbf{S}^{BV})_i - \frac{1}{2} \sum_i \lambda_i \{(\boldsymbol{\pi}^T \mathbf{S}^{BV})_i\}^2] . \tag{7}$$

Eq. (7) can be used to compute the sparse approximation within the sequential algorithm. We will only give a brief discussion here. In order to recompute the appropriate "cavity" parameters $\gamma^0_i$ and $\lambda^0_i$ when a new node is chosen by the algorithm, one removes a "pseudo-variable" $(\boldsymbol{\pi}^T \mathbf{S}_{BV})_i$ from the likelihood and recomputes the statistics of the remaining ones. When $i$ is in the BV set, then simply $(\boldsymbol{\pi}^T \mathbf{S}^{BV})_i = \mathbf{S}^{BV}_i$ and the computation reduces to the previous one. We will demonstrate the significance of this approach for two examples.

## 4.2 Independent Component Analysis

We consider a measured signal $\mathbf{X}_t$ which is assumed to be an instantaneous linear mixing of sources $\mathbf{S}$ corrupted with additive white Gaussian noise $\boldsymbol{\Gamma}$ that is,

$$\mathbf{X}_t = \mathbf{A} \mathbf{S}_t + \boldsymbol{\Gamma}_t , \tag{8}$$

where $\mathbf{A}$ is a (time independent) mixing matrix and the noise vector is assumed to be without temporal correlations having time independent covariance matrix $\mathbf{\Sigma}$. We thus have the following likelihood for parameters and sources at time $t$

$$P(\mathbf{X}_t|\mathbf{A}, \mathbf{\Sigma}, \mathbf{S}_t) = (\det 2\pi\mathbf{\Sigma})^{-\frac{1}{2}} e^{-\frac{1}{2}(\mathbf{X}_t - \mathbf{A}\mathbf{S}_t)^T \mathbf{\Sigma}^{-1}(\mathbf{X}_t - \mathbf{A}\mathbf{S}_t)} . \qquad (9)$$

and for all times $P(\mathbf{X}|\mathbf{A}, \mathbf{\Sigma}, \mathbf{S}) = \prod_t P(\mathbf{X}_t|\mathbf{A}, \mathbf{\Sigma}, \mathbf{S}_t)$. The aim of independent component analysis is to recover the unknown quantities: the sources $\mathbf{S}$, the mixing matrix $\mathbf{A}$ and the noise covariance $\mathbf{\Sigma}$ from the observed data using the assumption of statistical independence of the sources $P(\mathbf{S}_t) = \prod_i P(S_{it})$. Following [5], we estimate the mixing matrix $\mathbf{A}$ and the noise covariance $\mathbf{\Sigma}$, by an MLII procedure, i.e. by maximizing the Likelihood $P(\mathbf{X}|\mathbf{A}, \mathbf{\Sigma}) = \int d\mathbf{S} P(\mathbf{X}|\mathbf{A}, \mathbf{\Sigma}, \mathbf{S}) P(\mathbf{S})$. The corresponding estimates are $\mathbf{A}_{\mathrm{MLII}} = \sum_t \mathbf{X}_t \langle \mathbf{S}_t \rangle^T \left( \sum_{t'} \langle \mathbf{S}_{t'} \mathbf{S}_{t'}^T \rangle \right)^{-1}$ and $\mathbf{\Sigma}_{\mathrm{MLII}} = \frac{1}{N} \langle (\mathbf{X} - \mathbf{A}\mathbf{S})(\mathbf{X} - \mathbf{A}\mathbf{S})^T \rangle$. These estimates require averages over the posterior of $\mathbf{S}$ which has again the structure of the model eq. (1). They can be obtained efficiently using our sequential belief propagation algorithm in an iterative EM fashion, where the E-step amounts to estimating $\langle \mathbf{S}_t \rangle$ and $\langle \mathbf{S}_t \mathbf{S}_t^T \rangle$ with fixed $\mathbf{A}$ and $\mathbf{\Sigma}$ and the M-step consists of updating $\mathbf{A}$ and $\mathbf{\Sigma}$.

## 5 Simulations

### 5.1 Classification with GPs

This problem has been studied before [9, 4] using a sequential, sparse algorithm, based on a single sweep through the data only. Within the ADATAP approach we are able to perform multiple sweeps in order to achieve a self-consistent solution. The outputs are binary $y \in \{-1, 1\}$ and the likelihood is based on the probit model $P(y|\phi(x)) = \mathrm{Erf}(u) = \frac{1}{\sqrt{2\pi}} \int_{-\infty}^{u} dt \exp\left[-\frac{t^2}{2}\right]$. where $u = y\phi(x)/\sigma_0$ and $\sigma_0$ measures the noise level. The predictive distribution for a new test input $x$ is $\mathrm{Erf}(y\langle\phi(x)\rangle_t/\sigma_x)$ with $\sigma_x^2 = \sigma_0^2 + K_t(x, x)$, which is easily rewritten in terms of the parameters $\gamma$'s and $\chi$'s according to eqs. (5). We used the USPS dataset[1] of gray-scale handwritten digit images of size $16 \times 16$ with 7291 training patterns and 2007 test patterns. For the kernel we choose the RBF kernel $K_0(x, x') = a_K \exp(-\|x - x'\|^2/(m\sigma_K^2))$ where $m$ is the dimension of the inputs (256 in this case), and $a_K$ and $\sigma_K$ are parameters. In the simulations we used 7000 random training examples. We performed simulations for different sizes of the BV set and compared multiple iterations with a single sweep through the dataset. The results are displayed in Fig. 1. The lines show the average results of 5 runs where the task was to classify the digits into fours/non-fours. Our results show that, in contrast to the online learning, the fluctuations caused by the order of presentation are diminished (marked with bars on the figure).

### 5.2 Density estimation with GPs

Bayesian non-parametric models for density estimation can be defined [10] by parametrising densities $p$ as $p(x|\phi) = \frac{\phi^2(x)}{\int \phi^2(x)\,dx}$ and using a Gaussian process prior over the space of functions $\phi$. Observing $N$ data points $D = x_1, \ldots, x_N$, we can express the predictive distribution (again, $E$ denotes the expectation over the GP prior) as

$$
\begin{aligned}
p(x|D) &= \frac{1}{Z} E\left[ p(x|\phi) \prod_{i=1}^{N} p(x_i|\phi) \right] = \frac{1}{ZN!} \int_0^\infty dl\, l^N\, E\left[ \phi^2(x) \prod_{i=1}^{N} \phi^2(x_i)\, e^{-l \int \phi^2(x)dx} \right] \\
&\propto \int_0^\infty dl\, Z_l\, l^N\, E_l\left[ \phi^2(x) \prod_{i=1}^{N} \phi^2(x_i) \right] .
\end{aligned}
$$

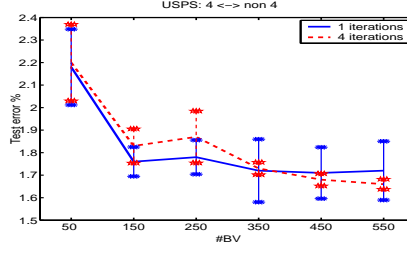

Figure 1: Results for classification for different BV sizes (x-axis) and multiple sweeps through the data.

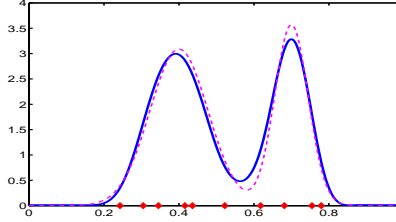

Figure 2: The GP estimation (continuous line) of a mixture of Gaussians (dotted line) using 10 BVs.

In the last expression, we have introduced an expectation over a new, effective Gaussian obtained by multiplying the old prior and the term $e^{-l \int \phi^2(x)dx}$ and normalizing by $Z_l$. We assume that for sufficiently large $N$ the integral over $l$ can be performed by Laplace's method, leaving us with an approximate predictor of the form $p(x|D) \propto \langle \phi^2(x) \rangle_l$, where the brackets denote posterior expectation for a GP model with a kernel that is a solution to the integral equation $K_l(x,y) = K_0(x,y) - l \int dz \ K_0(x,z)K_l(z,y)$. The likelihood of the fields $S_i \doteq \phi(x_i)$ at the observation points is $\rho_i(S) = S^2 e^{-\frac{1}{2}(K_l^{-1})_{ii}S^2}$. For any fixed $l$, we can apply the sparse ADATAP algorithm to this problem. After convergence of this inner loop, a new value of $l$ must be determined from (following a Laplace argument) $\frac{N}{l} = \langle \phi^2(x) \rangle_l$ until global convergence is achieved. To give a simplified toy example, we choose a kernel $K_0(x,y)$ which reproduces itself after convolution. Hence, the $l$ dependence is scaled out and we work with $l = 1$ and normalised at the end. We used a periodic kernel for data in $[0,1]$ given by

$$K_0(x,y) = -\cos(2\pi k_0(x-y)) + \sin(2\pi k_0(x-y))\cot(\pi(x-y)) \ .$$

$K_0$ has constant Fourier coefficients up to a cutoff frequency $k_0$ ($k_0 = 6$ in our simulations).

For the experiment we are using artificial data from a mixture of two Gaussians (dotted line in Fig. 2). We apply the sparse algorithm with multiple sweeps through the data. The sparsity also avoids the numerical problems caused by a possible close to singular Gram matrix. For the experiments, the size of the BV set was not limited a priori, and a similar criterion as in [4] was chosen in order to decide whether a data point should be included in the BV set or not. As a result, for $500$ training data, only $10$ were retained in the BV set. (continuous line in Fig. 2).

### 5.3 Independent Component Analysis

We have tested the sequential algorithm on an ICA problem for local feature extraction in hand written digits, i.e. extracting the different stroke styles [5] . We assumed positive components of $\mathbf{A}$ (enforced by Lagrange multipliers) and a positive prior

$$P(S_{it}) = \Theta(S_{it}) \exp(-S_{it}) \qquad (10)$$

As in [5] we used 500 handwritten '3's which are assumed to be generated by 25 hidden images. We compared a traditional parallel update algorithm with the sequential belief propagation algorithm. Both algorithms have computational complexity $\mathcal{O}(N^3)$. We find that the sequential algorithm needs only on average 7 sweeps through the sites to reach the desired accuracy whereas the parallel one fails to reach the desired accuracy in 100 sweeps using a somewhat larger number of flops. The adaptive TAP method using the sequential belief propagation approach is also not more computationally expensive than the linear response method used in [5].

## 6 Conclusion and Outlook

An obvious future direction for the ADATAP approach is the investigation of other minimization algorithms as an alternative to the EP approach outlined before. Also an extension of the sparse approximation to other non-GP models will be interesting. A highly important but difficult problem is the assessment of the accuracy of the approximation.

### Acknowledgments

M. Opper is grateful to Lars Kai Hansen for suggesting the non-parametric density model. O. Winther thanks Pedro Højen-Sørensen for the use of his Matlab code. The work is supported by EPSRC grant no. GR/M81601 and by the Danish Research Councils through Center for Biological Sequence Analysis.

## Footnotes

[1]Available from `http://www.kernel-machines.org/data/`

## References

[1] T.P. Minka. Expectation propagation for approximate Bayesian inference. PhD thesis, Dep. of Electrical Eng. and Comp. Sci.; MIT, 2000.

[2] J. S. Yedidia, W. T. Freeman and Y. Weiss, Generalized Belief Propagation, to appear in Advances in Neural Information Processing Systems (NIPS'2000), MIT Press (2001).

[3] M. Opper and O. Winther, Tractable approximations for probabilistic models: The adaptive TAP approach, Phys. Rev. Lett. **86**, 3695 (2001).

[4] L. Csató and M. Opper. Sparse Gaussian Processes. Neural Computation accepted (2001).

[5] P.A.d.F.R. Højen-Sørensen, O. Winther, and L. K. Hansen, Mean Field Approaches to Independent Component Analysis, Neural Computation accepted (2001). Available from http://www.cbs.dtu.dk/winther/

[6] T. Plefka, Convergence condition of the TAP equations for the infinite-ranged Ising spin glass model, J. Phys. A **15**, 1971 (1982).

[7] T. Tanaka, Mean-Field Theory of Boltzmann Machine Learning, Phys. Rev. E **58**, 2302(1998).

[8] G. Parisi and M. Potters, Mean-Field Equations for Spin Models with Orthogonal Interaction Matrices, J. Phys. A (Math. Gen.) **28**, 5267 (1995).

[9] L. Csató, E. Fokoué, M. Opper, B. Schottky, and O. Winther. Efficient approaches to Gaussian process classification. In *Advances in Neural Information Processing Systems*, volume 12, (2000).

[10] D.M. Schmidt. Continuous probability distributions from finite data. *arXiv:physics/9808005* (1998)
